# Learning as MAP Inference in Discrete Graphical Models

**Xianghang Liu**
NICTA/UNSW
Sydney, Australia
xianghang.liu@nicta.com.au

**James Petterson**
NICTA/ANU
Canberra, Australia
james.petterson@nicta.com.au

**Tiberio S. Caetano**
NICTA/ANU/University of Sydney
Canberra and Sydney, Australia
tiberio.caetano@nicta.com.au

## Abstract

We present a new formulation for binary classification. Instead of relying on convex losses and regularizers such as in SVMs, logistic regression and boosting, or instead non-convex but continuous formulations such as those encountered in neural networks and deep belief networks, our framework entails a non-convex but *discrete* formulation, where estimation amounts to finding a MAP configuration in a graphical model whose potential functions are low-dimensional discrete surrogates for the misclassification loss. We argue that such a discrete formulation can naturally account for a number of issues that are typically encountered in either the convex or the continuous non-convex approaches, or both. By reducing the learning problem to a MAP inference problem, we can immediately translate the guarantees available for many inference settings to the learning problem itself. We empirically demonstrate in a number of experiments that this approach is promising in dealing with issues such as severe label noise, while still having global optimality guarantees. Due to the discrete nature of the formulation, it also allows for *direct* regularization through cardinality-based penalties, such as the $\ell_0$ pseudo-norm, thus providing the ability to perform feature selection and trade-off interpretability and predictability in a principled manner. We also outline a number of open problems arising from the formulation.

## 1 Introduction

A large fraction of the machine learning community is concerned itself with the formulation of a learning problem as a single, well-defined *optimization* problem. This is the case for many popular techniques, including those associated with margin or likelihood-based estimators, such as SVMs, logistic regression, boosting, CRFs and deep belief networks. Among these optimization-based frameworks for learning, two paradigms stand out: the one based on convex formulations (such as SVMs) and the one based on non-convex formulations (such as deep belief networks). The main argument in favor of convex formulations is that we can effectively decouple modeling from optimization, what has substantial theoretical and practical benefits. In particular, it is of great value in terms of reproducibility, modularity and ease of use. Coming from the other end, the main argument for non-convexity is that a convex formulation very often fails to capture fundamental properties of a real problem (e.g. see [1, 2] for examples of some fundamental limitations of convex loss functions).

The motivation for this paper starts from the observation that the above tension is not really between convexity and non-convexity, but between convexity and *continuous* non-convexity. Historically, the optimization-based approach to machine learning has been virtually a synonym of *continuous* optimization. Estimation in continuous parameter spaces in some cases allows for closed-form solutions (such as in least-squares regression), or if not we can resort to computing gradients (for smooth continuous functions) or subgradients (for non-smooth continuous functions) which give us a generic tool for finding a local optimum of an arbitrary continuous function (global optimum if the continuous function is convex). On the contrary, unless P=NP there is no general tool to efficiently optimize discrete functions. We suspect this is one of the reasons why machine learning has traditionally been formulated in terms of continuous optimization: it is indeed convenient to compute gradients or subgradients and delegate optimization to some off-the-shelf gradient-based algorithm.

The formulation we introduce in this paper is non-convex, but *discrete* rather than continuous. By being non-convex we will attempt at capturing some of the expressive power of continuous non-convex formulations (such as robustness to labeling noise), and by being discrete we will retain the ability of convex formulations to provide theoretical guarantees in optimization. There are highly non-trivial classes of non-convex discrete functions defined over exponentially large discrete spaces which can be optimized efficiently. This is, after all, the main topic of combinatorial optimization. Discrete functions factored over cliques of low-treewidth graphs can be optimized efficiently via dynamic programming [3]. Arbitrary submodular functions can be minimized in polynomial time [4]. Particular submodular functions can be optimized very efficiently using max-flow algorithms [5]. Discrete functions defined over other particular classes of graphs also have polynomial-time algorithms (planar graphs [6], perfect graphs [7]). And of course although many discrete optimization problems are NP-hard, several have efficient constant-factor approximations [8]. In addition to all that, much progress has been done recently on developing tight LP relaxations for hard combinatorial problems [9]. Although all these discrete approaches have been widely used for solving *inference* problems in machine learning settings, we argue in this paper that they should also be used to solve *estimation* problems, or learning per se.

The discrete approach does pose several new questions though, which we list at the end. Our contribution is to outline the overall framework in terms of a few key ideas and assumptions, as well as to empirically evaluate in real-world datasets particular model instances within the framework. Although these instances are very simple, they already display important desirable behavior that is missing in state-of-the-art estimators such as SVMs.

## 2 Desiderata

We want to rethink the problem of learning a linear binary classifier. In this section we list the features that we would like a general-purpose learning machine for this problem to possess. These features essentially guide the assumptions behind our framework.

**Option to decouple modeling from optimization:** As discussed in the introduction, this is the great appeal of convex formulations, and we would like to retain it. Note however that we want the *option*, not necessarily a mandate of always decoupling modeling from optimization. We want to be able to please the user who is not an optimization expert or doesn't have the time or resources to refine the optimizer, by having the *option* of requesting the learning machine to configure itself in a mode in which global optimization is guaranteed and the runtime of optimization is precisely predictable. However we also want to please the user who *is* an expert, and is willing to spend a lot of time in refining the optimizer, to achieve the best possible results regardless of training time considerations. In our framework, we have the option to explore the spectrum between simpler models in which we can generate precise estimates of the runtime of the whole algorithm, and more complex models where we can focus on boosted performance at the expense of runtime predictability or demand for expert-exclusive fine-tuning skills.

**Option of Simplicity:** This point is related to the previous one, but it's more general. The complexity of a learning algorithm is a great barrier for its dissemination, even if it promises exceptional results once properly implemented. Most users of machine learning are not machine learning experts themselves, and for them in particular the cost of getting

a complex algorithm to work often outweighs the accuracy gains, especially if a reasonably good solution can be obtained with a very simple algorithm. For instance, in our framework the user has the option of reducing the learning algorithm to a series of matrix multiplications and lookup operations, while having a precise estimate of the total runtime of the algorithm and retaining good performance.

**Robustness to label noise:** SVMs are considered state-of-the-art estimators for binary classifiers, as well as boosting and logistic regression. All these optimize convex loss functions. However, when label noise is present, convex loss functions inflict arbitrarily large penalty on misclassifications because they are unbounded. In other words, in high label noise settings these convex loss functions become poor proxies for the 0/1 loss (the loss we really care about). This fundamental limitation of convex loss functions is well understood theoretically [1]. The fact that the loss function of interest is itself *discrete* is indeed a hint that maybe we should investigate *discrete surrogates* rather than continuous surrogates for the 0/1 loss: optimizing discrete functions over continuous spaces is hard, but not necessarily over discrete spaces. In our framework we directly address this issue.

**Ability to achieve sparsity:** Often we need to estimate sparse models. This can be for several reasons, including interpretability (be able to tell which are the 'most important' features), efficiency (at prediction time we can only afford to use a limited number of features) or, importantly, for purely statistical reasons (constraining the solution to low-dimensional subspaces has a regularization effect). The standard convex approach uses $\ell_1$ regularization. However the assumptions required to make $\ell_1$-regularized models be actually good proxies for the support cardinality function ($\ell_0$ pseudo-norm) are very strong and in practice rarely met [10]. In fact this has motivated an entire new line of work on structured sparsity, which tries to further regularize the solution so as to obtain better statistical properties in high dimensions [11, 12, 13]. This however comes at the price of more expensive optimization algorithms. Ideally we would like to regularize with $\ell_0$ *directly*; maybe this suggests the possibility of exploring an inherently discrete formulation? In our approach we have the ability to perform direct regularization via the $\ell_0$ pseudo-norm, or other scale-invariant regularizers.

**Leverage the power of low-dimensional approximations:** Machine learning folklore has it that the Naive Bayes assumption (features conditionally independent given the class label) often produces remarkably good classifiers. So a natural question is: is it really necessary to work directly in the original high-dimensional space, such as SVMs do? A key aspect of our framework is that we explicitly exploit the concept of composing a high-dimensional model from low-dimensional pieces. However we go beyond the Naive Bayes assumption by constructing *graphs* that model dependencies between variables. By varying the properties of these graphs we can trade-off model complexity and optimization efficiency in a straightforward manner.

## 3  Basic Setting

Much of current machine learning research studies estimators of the type

$$\underset{\theta \in \Theta}{\operatorname{argmin}} \sum_n \ell(y^n, f(x^n; \theta)) + \lambda \Omega(\theta) \tag{1}$$

where $\{x^n, y^n\}$ is a training set of inputs $x \in \mathcal{X}$ and outputs $y \in \mathcal{Y}$, assumed sampled independently from an unknown probability measure $P$ on $\mathcal{X} \times \mathcal{Y}$. $f : \mathcal{X} \to \mathcal{Y}$ is a member of a given class of predictors parameterized by $\theta$, $\Theta$ is a continuous space such as a Hilbert space, and $\ell$ as well as $\Omega$ are continuous and convex functions of $\theta$. $\ell$ is a loss function which enforces a penalty whenever $f(x^n) \neq y^n$, and therefore the first term in (1) measures the total loss incurred by predictor $f$ on the training sample $\{x^n, y^n\}$ under parameterization $\theta$. $\Omega$ controls the complexity of $\theta$ so as to avoid overfitting, and $\lambda$ trades-off the importance of a good fit to the training set versus model parsimony, so that good generalization is hopefully achieved.

Problem (1) is often called regularized empirical risk minimization, since the first term is the risk (expected loss) under the empirical distribution of the training data, and the second is a regularizer. This formulation is used for regression ($\mathcal{Y}$ continuous) as well as classification and structured prediction ($\mathcal{Y}$ discrete). Logistic Regression, Regularized Least-Squares

Regression, SVMs, CRFs, structured SVMs, Lasso, Group Lasso and a variety of other estimators are all instances of (1) for particular choices of $\ell$, $f$, $\Theta$ and $\Omega$. The formulation in (1) is a very general formulation for machine learning under the i.i.d. assumption.

In this paper we study problem (1) under the assumption that the parameter space $\Theta$ is discrete and finite, focusing on binary classification, when $\mathcal{Y} = \{-1, 1\}$.

## 4 Formulation

Our formulation departs from the one in (1) in two ways. The first assumption is that both the loss $\ell$ and the regularizer $\Omega$ are additive on low-dimensional functions defined by a graph $G = (V, E)$, i.e.,

$$\ell(y, f(x; \theta)) = \sum_{c \in \mathcal{C}} \ell_c(y, f_c(x; \theta_c)) \tag{2}$$

$$\Omega(\theta) = \sum_{c \in \mathcal{C}'} \Omega_c(\theta_c) \tag{3}$$

where $\mathcal{C} \cup \mathcal{C}'$ is the set of maximal cliques in $G$. Note that (3) is standard: $\ell_1$ and $\ell_2$ norms for example are both additive on singletons (in which case $\mathcal{C}' = V$). The arguably strong assumption here is (2). $\mathcal{C}$ is the set of parts where each part $c$ is, in principle, an arbitrary subset of $\{1, \ldots, D\}$, where $D$ is the dimensionality of the parameterization, i.e., $\theta = (\theta_1, \ldots, \theta_D)$. $\ell_c$ is a *low-dimensional discrete surrogate* for $\ell$, and $f_c$ is a *low-dimensional predictor*, both to be defined below. Note that in general two parameter subvectors $\theta_{c_i}$ and $\theta_{c_j}$ *are not independent* since the cliques $c_i$ and $c_j$ can overlap. Indeed, one of the key reasons sustaining the power of this formulation is that all $\theta_c$ are coupled either directly or indirectly through the connected graph $G = (V, E)$.

The second assumption is that $\Theta$ is discrete and therefore the vector $\theta = (\theta_1, \ldots, \theta_D)$ is discrete in the sense that $\theta_i$ is only allowed to take on finitely many values, including the value 0 (this will be important when we discuss regularization). For simplicity of exposition let's assume that the number of discrete values (bins) for each $\theta_i$ is the same: $B$. $B$ can be potentially quite large, for example it can be in the hundreds.

**Random Projections.** An instance $x$ above in reality is not the raw feature vector but instead a random projection of it into a space of the same or higher dimension, i.e., we effectively apply $X = RX'$ where $X'$ is the original data matrix, $R$ is a random matrix with entries drawn from $N(0, 1)$ and $X$ is the new data matrix. This often provides improved performance for our model due to the spreading of higher-order dependencies over lower-order cliques (when mapping to a higher dimensional space) and also is motivated from a theoretical argument (section 6). In what follows $x$ is the feature vector after the projection.

**Low-Dimensional Predictor.** We will assume a standard linear predictor of the kind

$$f_c(x; \theta) = \operatorname*{argmax}_{y \in \{-1, 1\}} y \langle x_c, \theta_c \rangle = \operatorname{sign} \langle x_c, \theta_c \rangle \tag{4}$$

In other words, we have a linear classifier that only considers the features in clique $c$.[1]

**Low-Dimensional Discrete Surrogates for the 0/1 loss** The low-dimensional discrete surrogate for the 0/1 loss is simply defined as the 0/1 loss incurred by predictor $f_c$:

$$\ell_c(y; f_c(x; \theta)) = (1 - y f_c(x; \theta))/2 \tag{5}$$

A key observation now is that $f_c$ and therefore $\ell_c$ can be computed in $O(B^k)$ by full enumeration over the $B^k$ instantiations of $\theta_c$, where $k$ is the size of clique $c$. In other words, the 0/1 loss constrained to the discretized subspace defined by clique $c$ can be exactly and efficiently computed (for small cliques).

**Regularization.** One critical technical issue is that linear predictors of the kind $\operatorname{argmax}_y \langle \phi(x, y), \theta \rangle$ are insensitive to scalings of $\theta$ [14]. Therefore, the loss $\ell$ will be such that $\ell(y, f(x; \alpha\theta)) = \ell(y, f(x; \theta))$ for $\alpha \neq 0$. This means that any regularizer that depends

on scale (such as $\ell_1$ and $\ell_2$ norms) is effectively meaningless since the minimization in (1) will drive $\Omega(\theta)$ to 0 (as this doesn't affect the loss). In other words, in such discrete setting we need a scale-invariant regularizer, such as the $\ell_0$ pseudo-norm. Note that $\ell_0$ is trivial to implement in this formulation, as we have enforced that the zero value must be included in the set of $B$ values attainable by each $\theta_i$:

$$\Omega(\theta) = \ell_0(\theta) = \sum_i \mathbf{1}_{\theta_i \neq 0} \qquad (6)$$

In addition, since this regularizer is additive on singletons $\theta_i$, it comes for free the fact that it does not contribute to the complexity of inference in the graphical model (i.e., it is a unary potential), which is a convenient property. Nothing prevents us however from having group regularizers, for example of the form $\sum_{c \in \mathcal{C}'} \lambda_c \mathbf{1}_{\theta_c \neq 0}$. Again, we can trade-off model simplicity and optimization efficiency by controlling the size of the maximal clique in $\mathcal{C}'$.

**Final optimization Problem.** After compiling the low-dimensional discrete proxies for the $0/1$ loss (the functions $l_c$) and incorporating our regularizer, we can assemble the following optimization problem

$$\operatorname*{argmin}_{\theta \in \Theta} \sum_{c \in \mathcal{C}} \underbrace{\sum_{n=1}^{N} \ell_c(y^n, f_c(x^n; \theta_c))}_{:= -N\psi_c(\theta_c)} + \sum_{i=1}^{D} \underbrace{\lambda \mathbf{1}_{\theta_i \neq 0}}_{:= -\lambda\phi_i(\theta_i)} \qquad (7)$$

which is a relaxation of (1) under all the above assumptions. The critical observation now is that (7) is a MAP inference problem in a discrete graphical model with clique set $\mathcal{C}$, high-order clique potentials $\psi_c(\theta_c)$ and unary potentials $\phi_i(\theta_i)$ [15]. Therefore we can resort to the vast literature on inference in graphical models to find exact or approximate solutions for (7). For example, if $G = (V, E)$ is a tree, then (7) can be solved exactly and efficiently using a dynamic programming algorithm that only requires matrix-vector multiplications in the $(min, +)$ semiring, in addition to elementary lookup operations [3]. For more general graphs the problem (7) can become NP-hard, but even in that case there are several principled approaches that often find excellent solutions, such as those based on linear programming relaxations [9] for tightly outer-bounding the marginal polytope [16]. In the experimental section we explore several options for constructing $G$, from simply generating a random chain (where MAP inference can be solved efficiently by dynamic programming) to generating dense random graphs (where MAP inference requires a more sophisticated approach such as an LP relaxation).

## 5 Related Work

The most closely related work we found is a recent paper by Potetz [17]. In a similar spirit to our approach, it also addresses the problem of estimating linear binary classifiers in a discrete formulation. However, instead of composing low-dimensional discrete surrogates of the $0/1$ loss as we do, it instead uses a fully connected factor graph and performs inference by estimating the mean of the max-marginals rather than MAP. Inference is approached using message-passing, which for the fully connected graph reduces to an intractable knapsack problem. In order to obtain a tractable model, the problem is then relaxed to a linear multiple choice knapsack problem, which can be solved efficiently. All the experiments though are performed on very low-dimensional datasets[2] and it is unclear how this approach would scale to high dimensionality while keeping a fully connected graph.

## 6 Analysis

Here we sketch arguments supporting the assumptions driving our formulation. Obtaining a rigorous theoretical analysis is left as an open problem for future research. Our assumptions involve three approximations of the problem of $0/1$ loss minimization. First, the discretization of the parameter space. Second, the computation of low-dimensional proxies for the $0/1$ loss rather than attacking the $0/1$ loss directly in the resulting discrete space. Finally, the use of a graph $G = (V, E)$ which in general will be *sparse*, i.e., not fully connected. We now discuss each of these assumptions.

## 6.1 Discretization of the parameter space

The explicit enforcement of a finite number of possible values for each parameter may seem at first a strong assumption. However, a key observation here is that we are restricting ourselves to *linear predictors*, which basically means that, for any sample, small perturbations of a random hyperplane will with high probability induce at most small changes in the 0/1 loss. Therefore there are good reasons to believe that indeed, for linear predictors, increasing binning has a diminishing returns behavior and after only a moderate amount of bins no much improvement can be obtained. This assumption is also used in [17].

## 6.2 Low-dimensional proxies for the 0/1 loss

This assumption can be justified using recent results stating that the margin is well-preserved under random projections to low-dimensional subspaces [18, 19]. For instance, Theorem 6 in [19] shows that the margin is preserved with high probability for embeddings with dimension only logarithmic on the sample size (a result similar in spirit to the Johnson-Lindenstrauss Lemma [20]). Since the (soft)margin upper bounds the 0/1 loss, this should also be preserved with at least equivalent guarantees.

## 6.3 Graph sparsity

This is apparently the strongest assumption. In our formulation, we impose conditional independence assumptions on the set of random variables used as features. There are two main observations. The first is that in real high-dimensional data the existence of (approximate) conditional independences is more of a rule than an exception. This is directly related to the fact that usually high-dimensional data inhabit low-dimensional manifolds or subspaces. In our case, we have a graph with the nodes representing different features, and this can be seen as a patching of low-dimensional subspaces, where each subspace is defined by one of the cliques in the graph. We do not address in this work how to optimally determine a subgraph, leaving that as an open problem in this framework. Rather, we show that even with *random* subgraphs, and in particular subgraphs as simple as *chains*, we can obtain models that have high accuracy and remarkable robustness to high degrees of label noise. The second observation is that nothing prevents us from using quite dense graphs and seeking approximate rather than exact MAP inference, say through LP relaxations [9]. Indeed we illustrate this possibility in the experimental section below.

## 7 Experiments

**Settings.** To evaluate our method (DISCRETE) for binary classification problems, we apply it to real-world datasets and compared it to linear Support Vector Machines (SVM), which are a state-of-the-art estimator for linear classifiers. We note that although both use linear predictors, the model classes are not identical: since we use discretization, the set of hyperplanes our estimator will optimize over is strictly smaller. We run these algorithms on publicly available datasets from the UCI machine learning repository [21]. See Table 1 for the details of these datasets. For both algorithms, the only hyperparameter is the trade-off between the loss and the regularization term. We run 5-fold cross validation for both methods to select the optimal hyperparameters. The number of bins used for discretization may affect the accuracy of DISCRETE. For the experiments, we fix it to 11, since for larger values there was negligible improvement (which supports our argument from section 6.1).

**Robustness to Label Noise.** In the first experiment, we test the robustness of different methods to increasing label noise. We first flip the labels of the training data with increasing probability from 0 to 0.4 and then run these algorithms on the noisy training data. The plots of the classification accuracy at each noise level are shown in Figure 1. For DISCRETE, we used as the graph $G$ a *random chain*, i.e., the simplest possible option for a connected graph. In this case, optimization is straightforward via a Viterbi algorithm: a sequence of matrix-vector multiplications in the $(min, +)$ semiring with trivial bookkeeping and subsequential lookup, which will run in $O(B^2D)$ since we have $B$ states per variable and $D$ variables. To assess the effect of randomization, we run on 20 random chains and plot both the average and the standard error obtained. The impact of randomization seems negligible. From Figure 1, DISCRETE demonstrates classification accuracy only slightly inferior to SVM in

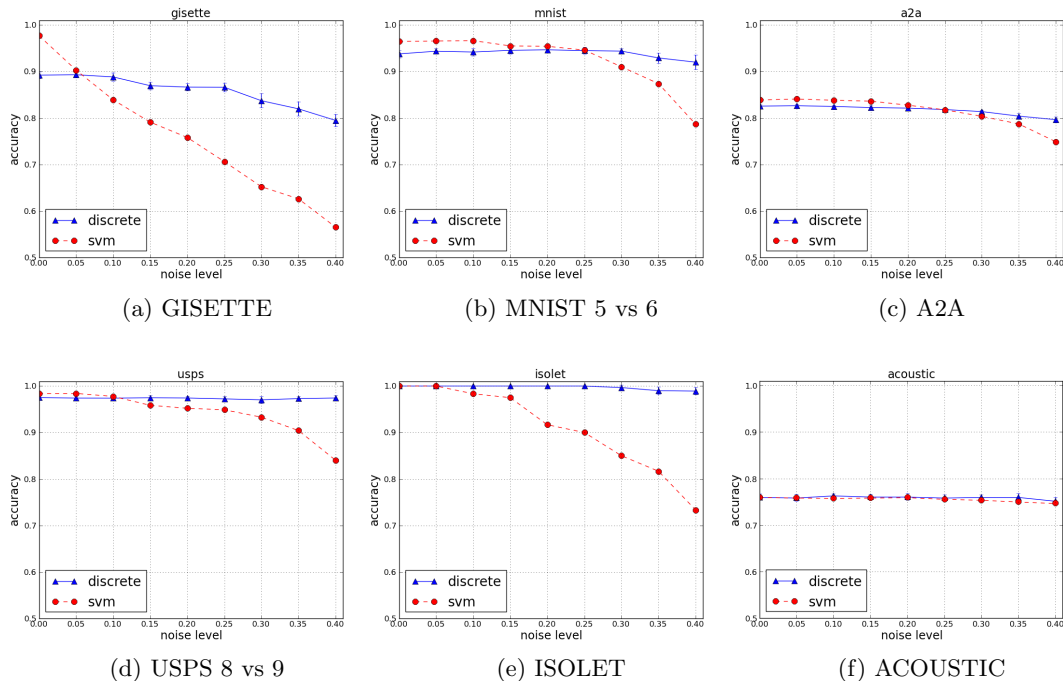

Figure 1: Comparison of the Discrete Method and Linear SVM

the noiseless regime (i.e., when the hinge loss is a good proxy for the 0/1 loss). However, as soon as a significant amount of label noise is present, SVM degrades substantially while DISCRETE remains remarkably stable, delivering high accuracy even after flipping labels with 40% probability. We believe these are significant results given the truly elementary nature of the optimization procedure: the method is simple, fast and the runtime can be predicted with high accuracy since there is a determined number of operations; $2(D-1)$ messages are passed, each with worst-case runtime of $O(B^2)$ determined by the matrix-vector multiplication. Note in particular how this differs from continuous optimization settings in which the analysis is in terms of rate of convergence rather than the precise number of discrete operations performed. It is also interesting to observe that for different values of the cross-validation parameter our algorithm runs in precisely the same amount of time, while for SVMs convergence will be much slower for small scalings of the regularizer since the relative importance of the non-differentiable hinge loss over the strongly convex quadratic term increases. This experiment shows that even if we have the simplest setting of our formulation (random chains, which comes with very fast and exact MAP inference) we can still obtain results that are close or similar to those obtained by the state-of-the-art linear SVM classifier in the noiseless case, and superior for high levels of label noise.

**Evaluation without Noise.** As seen in Figure 1, in the noiseless (or small noise) regime SVM is often slightly superior to our random chain model. A natural question to ask is therefore how would more complex graph topologies perform. Here we run experiments on two other types of graphs: a random 2-chain (i.e. a random junction tree with cliques $\{i, i+1, i+2\}$) and a random $k$-regular graph, where $k$ is set to be such that the resulting graph has 10% of the possible edges. For the 2-chain, the optimization algorithm is exact inference via $(min, +)$ message-passing, just as the Viterbi algorithm, but now applied to a larger clique, which increases the memory and runtime cost by $O(B)$. For the random graph, we obtain a more complex topology in which exact inference is intractable. In our experiments we used the approximate inference algorithm from [22], which solves optimally and efficiently an LP relaxation via the alternating direction method of multipliers, ADMM [23].

Table 1: Datasets used for the experiments in Figure 1

|  | GISETTE | MNIST | A2A | USPS | ISOLET | ACOUSTIC |
|---|---|---|---|---|---|---|
| # Train | 6000 | 10205 | 2265 | 950 | 480 | 19705 |
| # Test | 1000 | 1134 | 30296 | 237 | 120 | 78823 |
| # Features | 5000 | 784 | 123 | 256 | 617 | 50 |

Table 2: Error rates of different methods for binary classification, *without label noise*. In this setting, the hinge loss used by SVM is an excellent proxy for the 0/1 loss. Yet, the proposed variants (top 3 rows) are still competitive in most datasets.

|  | GISETTE | MNIST | A2A | USPS | ISOLET | ACOUSTIC |
|---|---|---|---|---|---|---|
| random chain | 89.23 | 93.79 | 82.55 | 97.51 | 100 | 76.01 |
| random 2-chain | 89 | 94.47 | 82.65 | 97.78 | 100 | 76.55 |
| random graph | 88.6 | 94.89 | 83.17 | 97.44 | 100 | 74.80 |
| SVM | 97.7 | 96.47 | 83.88 | 98.4 | 100 | 76.01 |

# 8    Extensions and Open Problems

Clearly the results in this paper are only a first step in the direction proposed. Several questions arise from this formulation.

**Theory.** In section 6 we only sketched the reasons why we pursued the assumptions laid out in this paper. We did not present any rigorous quantitative arguments analyzing the limitations of our formulation. This is left as an open problem. However we believe section 6 does point to the key ideas that will ultimately underly a quantitative theory.

**Extension to multi-class and structured prediction.** In this work we only study binary classification problems. The extension to multi-class and structured prediction, as well as other learning settings is an open problem.

**Adaptive binning.** When discretizing the parameters, we used a fixed number of bins. This can be made more elaborate through the use of adaptive binning techniques that are dependent on the information content of each variable.

**Informative graph construction.** We only explored randomly generated graphs. The problem of selecting a graph topology in an informative way is highly relevant and is left open. For example B-matching can be used to generate an informative regular graph [24]. This problem is essentially a manifold learning problem and there are several ways it could be approached. Existing work on supervised manifold learning is very relevant here.

**Nonparametric extension.** We considered only linear parametric models. It would be interesting to consider nonparametric models, where the discretization occurs at the level of parameters associated with each training instance (as in the dual formulation of SVMs).

# 9    Conclusion

We presented a discrete formulation for learning linear binary classifiers. Parameters associated with features of the linear model are discretized into bins, and low-dimensional discrete surrogates of the 0/1 loss restricted to small groups of features are constructed. This results in a data structure that can be seen as a graphical model, where regularized risk minimization can be performed via MAP inference. We sketch theoretical arguments supporting the assumptions underlying our proposal and present empirical evidence that very simple, easily and quickly trainable models estimated with such a procedure can deliver results that are often comparable to those obtained by linear SVMs for noiseless scenarios, and superior under moderate to severe label noise.

# Acknowledgements

We thank E. Bonilla, A. Defazio, D. García-García, S. Gould, J. McAuley, S. Nowozin, M. Reid, S. Sanner and B. Williamson for discussions. NICTA is funded by the Australian Government as represented by the Department of Broadband, Communications and the Digital Economy and the Australian Research Council through the ICT Centre of Excellence program.

## Footnotes

[1]For notational simplicity we assume an offset parameter is already included in $\theta_c$ and a corresponding entry of 1 is appended to the vector $x_c$.

[2]Seven datasets with dimensionalities 7,9,10,11,14,15 and 61. See [17].

# References

[1] P. M. Long and R. A. Servedio, "Random classification noise defeats all convex potential boosters," *Machine Learning*, vol. 78, no. 3, pp. 287–304, 2010.

[2] P. M. Long and R. A. Servedio, "Learning large-margin halfspaces with more malicious noise," in *NIPS*, 2011.

[3] S. M. Aji and R. J. McEliece, "The generalized distributive law," *IEEE Trans. Inform. Theory*, vol. 46, no. 2, pp. 325–343, 2000.

[4] B. Korte and J. Vygen, *Combinatorial Optimization: Theory and Algorithms*. Springer Publishing Company, Incorporated, 4th ed., 2007.

[5] V. Kolmogorov and R. Zabih, "What energy functions can be minimizedvia graph cuts?," *IEEE Transactions on Pattern Analysis and Machine Intelligence*, vol. 26, pp. 147–159, 2004.

[6] A. Globerson and T. S. Jaakkola, "Approximate inference using planar graph decomposition," in *Advances in Neural Information Processing Systems 19* (B. Schölkopf, J. Platt, and T. Hoffman, eds.), pp. 473–480, Cambridge, MA: MIT Press, 2007.

[7] T. Jebara, "Perfect graphs and graphical modeling." To Appear in Tractability, Cambridge University Press, 2012.

[8] V. V. Vazirani, *Approximation Algorithms*. Springer, 2004.

[9] D. Sontag, *Approximate Inference in Graphical Models using LP Relaxations*. PhD thesis, Massachusetts Institute of Technology, Department of Electrical Engineering and Computer Science, 2010.

[10] P. Zhao and B. Yu, "On model selection consistency of lasso," *J. Mach. Learn. Res.*, vol. 7, pp. 2541–2563, Dec. 2006.

[11] F. Bach, R. Jenatton, J. Mairal, and G. Obozinski, "Structured sparsity through convex optimization." Technical report, HAL 00621245-v2, to appear in Statistical Science, 2012.

[12] J. Huang, T. Zhang, and D. Metaxas, "Learning with structured sparsity," in *Proceedings of the 26th Annual International Conference on Machine Learning*, ICML '09, (New York, NY, USA), pp. 417–424, ACM, 2009.

[13] F. R. Bach, "Structured sparsity-inducing norms through submodular functions," in *NIPS*, pp. 118–126, 2010.

[14] D. Mcallester and J. Keshet, "Generalization bounds and consistency for latent structural probit and ramp loss," in *Advances in Neural Information Processing Systems 24* (J. Shawe-Taylor, R. Zemel, P. Bartlett, F. Pereira, and K. Weinberger, eds.), pp. 2205–2212, 2011.

[15] D. Koller and N. Friedman, *Probabilistic Graphical Models: Principles and Techniques*. MIT Press, 2009.

[16] M. J. Wainwright and M. I. Jordan, *Graphical Models, Exponential Families, and Variational Inference*. Hanover, MA, USA: Now Publishers Inc., 2008.

[17] B. Potetz, "Estimating the bayes point using linear knapsack problems," in *ICML*, pp. 257–264, 2011.

[18] M.-F. Balcan, A. Blum, and S. Vempala, "Kernels as features: On kernels, margins, and low-dimensional mappings," *Machine Learning*, vol. 65, no. 1, pp. 79–94, 2006.

[19] Q. Shi, C. Chen, R. Hill, and A. van den Hengel, "Is margin preserved after random projection?," in *ICML*, 2012.

[20] S. Dasgupta and A. Gupta, "An elementary proof of a theorem of johnson and lindenstrauss," *Random Struct. Algorithms*, vol. 22, pp. 60–65, Jan. 2003.

[21] A. Frank and A. Asuncion, "UCI machine learning repository," 2010.

[22] O. Meshi and A. Globerson, "An alternating direction method for dual map lp relaxation," in *Proceedings of the 2011 European conference on Machine learning and knowledge discovery in databases - Volume Part II*, ECML PKDD'11, (Berlin, Heidelberg), pp. 470–483, Springer-Verlag, 2011.

[23] S. Boyd, N. Parikh, E. Chu, B. Peleato, and J. Eckstein, "Distributed optimization and statistical learning via the alternating direction method of multipliers," *Foundations and Trends in Machine Learning*, vol. 3, 2011.

[24] T. Jebara, J. Wang, and S. Chang, "Graph construction and b-matching for semi-supervised learning," in *ICML*, 2009.

